# Hidden Markov Model Induction by Bayesian Model Merging

**Andreas Stolcke**[*,**]
[*]Computer Science Division
University of California
Berkeley, CA 94720
stolcke@icsi.berkeley.edu

**Stephen Omohundro**[**]
[**]International Computer Science Institute
1947 Center Street, Suite 600
Berkeley, CA 94704
om@icsi.berkeley.edu

## Abstract

This paper describes a technique for learning both the number of states and the topology of Hidden Markov Models from examples. The induction process starts with the most specific model consistent with the training data and generalizes by successively merging states. Both the choice of states to merge and the stopping criterion are guided by the Bayesian posterior probability. We compare our algorithm with the Baum-Welch method of estimating fixed-size models, and find that it can induce minimal HMMs from data in cases where fixed estimation does not converge or requires redundant parameters to converge.

## 1  INTRODUCTION AND OVERVIEW

Hidden Markov Models (HMMs) are a well-studied approach to the modelling of sequence data. HMMs can be viewed as a stochastic generalization of finite-state automata, where both the transitions between states and the generation of output symbols are governed by probability distributions. HMMs have been important in speech recognition (Rabiner & Juang, 1986), cryptography, and more recently in other areas such as protein classification and alignment (Haussler, Krogh, Mian & Sjölander, 1992; Baldi, Chauvin, Hunkapiller & McClure, 1993).

Practitioners have typically chosen the HMM topology by hand, so that learning the HMM from sample data means estimating only a fixed number of model parameters. The standard approach is to find a maximum likelihood (ML) or maximum *a posteriori* probability (MAP) estimate of the HMM parameters. The Baum-Welch algorithm uses dynamic programming

to approximate these estimates (Baum, Petrie, Soules & Weiss, 1970).

A more general problem is to additionally find the best HMM topology. This includes both the number of states and the connectivity (the non-zero transitions and emissions). One could exhaustively search the model space using the Baum-Welch algorithm on fully connected models of varying sizes, picking the model size and topology with the highest posterior probability. (Maximum likelihood estimation is not useful for this comparison since larger models usually fit the data better.) This approach is very costly and Baum-Welch may get stuck at sub-optimal local maxima. Our comparative results later in the paper show that this often occurs in practice. The problem can be somewhat alleviated by sampling from several initial conditions, but at a further increase in computational cost.

The HMM induction method proposed in this paper tackles the structure learning problem in an incremental way. Rather than estimating a fixed-size model from scratch for various sizes, the model size is adjusted as new evidence arrives. There are two opposing tendencies in adjusting the model size and structure. Initially new data adds to the model size, because the HMM has to be augmented to accommodate the new samples. If enough data of a similar structure is available, however, the algorithm collapses the shared structure, decreasing the model size. The merging of structure is also what drives generalization, i.e., creates HMMs that generate data not seen during training.

Beyond being incremental, our algorithm is data-driven, in that the samples themselves completely determine the initial model shape. Baum-Welch estimation, by comparison, uses an initially random set of parameters for a given-sized HMM and iteratively updates them until a point is found at which the sample likelihood is locally maximal. What seems intuitively troublesome with this approach is that the initial model is completely uninformed by the data. The sample data directs the model formation process only in an indirect manner as the model approaches a meaningful shape.

## 2    HIDDEN MARKOV MODELS

For lack of space we cannot give a full introduction to HMMs here; see Rabiner & Juang (1986) for details. Briefly, an HMM consists of states and transitions like a Markov chain. In the discrete version considered here, it generates strings by performing random walks between an initial and a final state, outputting symbols at every state in between. The probability $P(x|M)$ that a model $M$ generates a string $x$ is determined by the conditional probabilities of making a transition from one state to another and the probability of emitting each symbol from each state. Once these are given, the probability of a particular path through the model generating the string can be computed as the product of all transition and emission probabilities along the path. The probability of a string $x$ is the sum of the probabilities of all paths generating $x$.

For example, the model $M_3$ in Figure 1 generates the strings $ab, abab, ababab, \ldots$ with probabilities $\frac{2}{3}, \frac{2}{3^2}, \frac{2}{3^3}, \ldots$, respectively.

## 3    HMM INDUCTION BY STATE MERGING

### 3.1    MODEL MERGING

Omohundro (1992) has proposed an approach to statistical model inference in which initial

models simply replicate the data and generalize by similarity. As more data is received, component models are fit from more complex model spaces. This allows the formation of arbitrarily complex models without overfitting along the way. The elementary step used in modifying the overall model is a *merging* of sub-models, collapsing the sample sets for the corresponding sample regions. The search for sub-models to merge is guided by an attempt to sacrifice as little of the sample likelihood as possible as a result of the merging process. This search can be done very efficiently if (a) a greedy search strategy can be used, and (b) likelihood computations can be done locally for each sub-model and don't require global recomputation on each model update.

## 3.2   STATE MERGING IN HMMS

We have applied this general approach to the HMM learning task. We describe the algorithm here mostly by presenting an example. The details are available in Stolcke & Omohundro (1993).

To obtain an initial model from the data, we first construct an HMM which produces exactly the input strings. The start state has as many outgoing transitions as there are strings and each string is represented by a unique path with one state per sample symbol. The probability of entering these paths from the start state is uniformly distributed. Within each path there is a unique transition arc whose probability is 1. The emission probabilities are 1 for each state to produce the corresponding symbol.

As an example, consider the regular language $(ab)^+$ and two samples drawn from it, the strings *ab* and *abab*. The algorithm constructs the initial model $M_0$ depicted in Figure 1. This is the most specific model accounting for the observed data. It assigns each sample a probability equal to its relative frequency, and is therefore a maximum likelihood model for the data.

Learning from the sample data means generalizing from it. This implies trading off model likelihood against some sort of bias towards 'simpler' models, expressed by a prior probability distribution over HMMs. Bayesian analysis provides a formal basis for this tradeoff. Bayes' rule tells us that the posterior model probability $P(M|x)$ is proportional to the product of the model prior $P(M)$ and the likelihood of the data $P(x|M)$. Smaller or simpler models will have a higher prior and this can outweigh the drop in likelihood as long as the generalization is conservative and keeps the model close to the data. The choice of model priors is discussed in the next section.

The fundamental idea exploited here is that the initial model $M_0$ can be gradually transformed into the generating model by repeatedly *merging states*. The intuition for this heuristic comes from the fact that if we take the paths that generate the samples in an actual generating HMM $M$ and 'unroll' them to make them completely disjoint, we obtain $M_0$. The iterative merging process, then, is an attempt to undo the unrolling, tracing a search through the model space back to the generating model.

Merging two states $q_1$ and $q_2$ in this context means replacing $q_1$ and $q_2$ by a new state $r$ with a transition distribution that is a weighted mixture of the transition probabilities of $q_1$, $q_2$, and with a similar mixture distribution for the emissions. Transition probabilities into $q_1$ or $q_2$ are added up and redirected to $r$. The weights used in forming the mixture distributions are the relative frequencies with which $q_1$ and $q_2$ are visited in the current model.

Repeatedly performing such merging operations yields a sequence of models $M_0$, $M_1$,

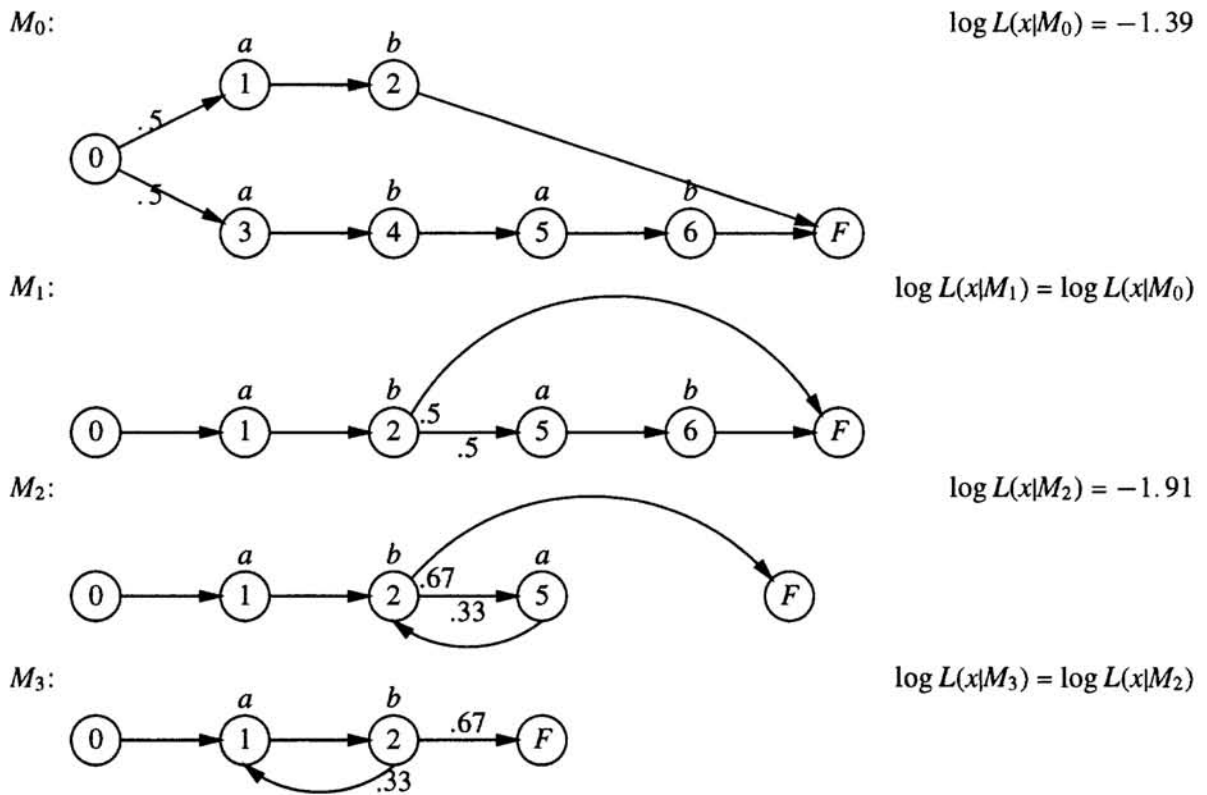

Figure 1: Sequence of models obtained by merging samples $\{ab, abab\}$. All transitions without special annotations have probability 1; Output symbols appear above their respective states and also carry an implicit probability of 1. For each model the log likelihood is given.

$M_2, ...,$ along which we can search for the MAP model. To make the search for $M$ efficient, we use a greedy strategy: given $M_i$, choose a pair of states for merging that maximizes $P(M_{i+1}|X)$.

Continuing with the previous example, we find that states 1 and 3 in $M_0$ can be merged without penalizing the likelihood. This is because they have identical outputs and the loss due to merging the outgoing transitions is compensated by the merging of the incoming transitions. The .5/.5 split is simply transferred to outgoing transitions of the merged state. The same situation obtains for states 2 and 4 once 1 and 3 are merged. From these two first merges we get model $M_1$ in Figure 1. By convention we reuse the smaller of two state indices to denote the merged state.

At this point the best merge turns out to be between states 2 and 6, giving model $M_2$. However, there is a penalty in likelihood, which decreases to about .59 of its previous value. Under all the reasonable priors we considered (see below), the posterior model probability still increases due to an increase in the prior. Note that the transition probability ratio at state 2 is now 2/1, since two samples make use of the first transition, whereas only one takes the second transition.

Finally, states 1 and 5 can be merged without penalty to give $M_3$, the minimal model that generates $(ab)^+$. Further merging at this point would reduce the likelihood by three orders of magnitude. The resulting decrease in the posterior probability tells the algorithm to stop

at this point.

## 3.3 MODEL PRIORS

As noted previously, the likelihoods $P(X|M_i)$ along the sequence of models considered by the algorithm is monotonically decreasing. The prior $P(M)$ must account for an overall increase in posterior probability, and is therefore the driving force behind generalization.

As in the work on Bayesian learning of classification trees by Buntine (1992), we can split the prior $P(M)$ into a term accounting for the model structure, $P(M_s)$, and a term for the adjustable parameters in a fixed structure $P(M_p|M_s)$.

We initially relied on the structural prior only, incorporating an explicit bias towards smaller models. Size here is some function of the number of states and/or transitions, $|M|$. Such a prior can be obtained by making $P(M_s) \propto e^{-|M|}$, and can be viewed as a *description length prior* that penalizes models according to their coding length (Rissanen, 1983; Wallace & Freeman, 1987). The constants in this "MDL" term had to be adjusted by hand from examples of 'desirable' generalization.

For the parameter prior $P(M_p|M_s)$, it is standard practice to apply some sort of smoothing or regularizing prior to avoid overfitting the model parameters. Since both the transition and the emission probabilities are given by multinomial distributions it is natural to use a Dirichlet conjugate prior in this case (Berger, 1985). The effect of this prior is equivalent to having a number of 'virtual' samples for each of the possible transitions and emissions which are added to the actual samples when it comes to estimating the most likely parameter settings. In our case, the virtual samples made equal use of all potential transitions and emissions, adding bias towards uniform transition and emission probabilities.

We found that the Dirichlet priors by themselves produce an implicit bias towards smaller models, a phenomenon that can be explained as follows. The prior alone results in a model with uniform, flat distributions. Adding actual samples has the effect of putting bumps into the posterior distributions, so as to fit the data. The more samples are available, the more peaked the posteriors will get around the maximum likelihood estimates of the parameters, increasing the MAP value. In estimating HMM parameters, what counts is not the total number of samples, but the number of samples *per state*, since transition and emission distributions are local to each state. As we merge states, the available evidence gets shared by fewer states, thus allowing the remaining states to produce a better fit to the data.

This phenomenon is similar, but not identical, to the Bayesian 'Occam factors' that prefer models with fewer parameter (MacKay, 1992). Occam factors are a result of integrating the posterior over the parameter space, something which we do not do because of the computational complications it introduces in HMMs (see below).

## 3.4 APPROXIMATIONS

At each iteration step, our algorithm evaluates the posterior resulting from every possible merge in the current HMM. To keep this procedure feasible, a number of approximations are incorporated in the implementation that don't seem to affect its qualitative properties.

- For the purpose of likelihood computation, we consider only the most likely path through the model for a given sample string (the Viterbi path). This allows us to

express the likelihood in product form, computable from sufficient statistics for each transition and emission.

- We assume the Viterbi paths are preserved by the merging operation, that is, the paths previously passing through the merged states now go through the resulting new state. This allows us to update the sufficient statistics incrementally, and means only $O$(number of states) likelihood terms need to be recomputed.

- The posterior probability of the model structure is approximated by the posterior of the MAP estimates for the model parameters. Rigorously integrating over all parameter values is not feasible since varying even a single parameter could change the paths of all samples through the HMM.

- Finally, it has to be kept in mind that our search procedure along the sequence of merged models finds only local optima, since we stop as soon as the posterior starts to decrease. A full search of the space would be much more costly. However, we found a *best-first look-ahead* strategy to be sufficient in rare cases where a local maximum caused a problem. In those cases we continue merging along the best-first path for a fixed number of steps (typically one) to check whether the posterior has undergone just a temporary decrease.

## 4   EXPERIMENTS

We have used various artificial finite-state languages to test our algorithm and compare its performance to the standard Baum-Welch algorithm.

Table 1 summarizes the results on the two sample languages $ac^*a \cup bc^*b$ and $a^+b^+a^+b^+$. The first of these contains a contingency between initial and final symbols that can be hard for learning algorithms to uncover.

We used no explicit model size prior in our experiments after we found that the Dirichlet prior was very robust in giving just the the right amount of bias toward smaller models.[1] Summarizing the results, we found that merging very reliably found the generating model structure from a very small number of samples. The parameter values are determined by the sample set statistics.

The Baum-Welch algorithm, much like a backpropagation network, may be sensitive to its random initial parameter settings. We therefore sampled from a number of initial conditions. Interestingly, we found that Baum-Welch has a good chance of settling into a suboptimal HMM structure, especially if the number of states is the minimal number required for the target language. It proved much easier to estimate correct language models when extra states were provided. Also, increasing the sample size helped it converge to the target model.

## 5   RELATED WORK

Our approach is related to several other approaches in the literature.

The concept of state merging is implicit in the notion of state equivalence classes, which is fundamental to much of automata theory (Hopcroft & Ullman, 1979) and has been applied

| (a) | Method | Sample | Entropy | Cross-entropy | Language | $n$ |
|---|---|---|---|---|---|---|
| | Merging | 8 m.p. | 2.295 | $2.188 \pm .020$ | $ac^*a \cup bc^*b$ | 6 |
| | Merging | 20 random | 2.087 | $2.158 \pm .033$ | $ac^*a \cup bc^*b$ | 6 |
| | Baum-Welch | 8 m.p. | 2.087 | $2.894 \pm .023$ (best) | $(a \cup b)c^*(a \cup b)$ | 6 |
| | (10 trials) | | 2.773 | $4.291 \pm .228$ (worst) | $(a \cup b)c^*(a \cup b)$ | 6 |
| | Baum-Welch | 20 random | 2.087 | $2.105 \pm .031$ (best) | $ac^*a \cup bc^*b$ | 6 |
| | (10 trials) | | 2.775 | $2.825 \pm .031$ (worst) | $(a \cup b)c^*(a \cup b)$ | 6 |
| | Baum-Welch | 8 m.p. | 2.384 | $3.914 \pm .271$ | $ac^*a \cup bc^*b$ | 10 |
| | Baum-Welch | 20 random | 2.085 | $2.155 \pm .032$ | $ac^*a \cup bc^*b$ | 10 |

| (b) | Method | Sample | Entropy | Cross-entropy | Language | $n$ |
|---|---|---|---|---|---|---|
| | Merging | 5 m.p. | 2.163 | $7.678 \pm .158$ | $a^+b^+a^+b^+$ | 4 |
| | Baum-Welch | 5 m.p. | 3.545 | $8.963 \pm .161$ (best) | $(a^+b^+)^+$ | 4 |
| | (3 trials) | | 3.287 | $59.663 \pm .007$ (worst) | $(a^+b^+)^+$ | 4 |
| | Merging | 10 random | 5.009 | $5.623 \pm .074$ | $a^+b^+a^+b^+$ | 4 |
| | Baum-Welch | 10 random | 5.009 | $5.688 \pm .076$ (best) | $a^+b^+a^+b^+$ | 4 |
| | (3 trials) | | 6.109 | $8.395 \pm .137$ (worst) | $(a^+b^+)^+$ | 4 |

Table 1: Results for merging and Baum-Welch on two regular languages: (a) $ac^*a \cup bc^*b$ and (b) $a^+b^+a^+b^+$. Samples were either the top most probable (m.p.) ones from the target language, or a set of randomly generated ones. 'Entropy' is the average negative log probability on the training set, whereas 'cross-entropy' refers to the empirical cross-entropy between the induced model and the generating model (the lower, the better generalization). $n$ denotes the final number of model states for merging, or the fixed model size for Baum-Welch. For Baum-Welch, both best and worst performance over several initial conditions is listed.

to automata learning as well (Angluin & Smith, 1983).

Tomita (1982) is an example of finite-state model space search guided by a (non-probabilistic) goodness measure.

Horning (1969) describes a Bayesian grammar induction procedure that searches the model space exhaustively for the MAP model. The procedure provably finds the globally optimal grammar in finite time, but is infeasible in practice because of its enumerative character.

The incremental augmentation of the HMM by merging in new samples has some of the flavor of the algorithm used by Porat & Feldman (1991) to induce a finite-state model from positive-only, ordered examples.

Haussler et al. (1992) use limited HMM 'surgery' (insertions and deletions in a linear HMM) to adjust the model size to the data, while keeping the topology unchanged.

# 6   FURTHER RESEARCH

We are investigating several real-world applications for our method. One task is the construction of unified multiple-pronunciation word models for speech recognition. This is currently being carried out in collaboration with Chuck Wooters at ICSI, and it appears that our merging algorithm is able to produce linguistically adequate phonetic models.

Another direction involves an extension of the model space to stochastic context-free grammars, for which a standard estimation method analogous to Baum-Welch exists (Lari

& Young, 1990). The notions of sample incorporation and merging carry over to this domain (with merging now involving the non-terminals of the CFG), but need to be complemented with a mechanism that adds new non-terminals to create hierarchical structure (which we call chunking).

## Acknowledgements

We would like to thank Peter Cheeseman, Wray Buntine, David Stoutamire, and Jerry Feldman for helpful discussions of the issues in this paper.

## Footnotes

[1]The number of 'virtual' samples per transition/emission was held constant at 0.1 throughout.

# References

Angluin, D. & Smith, C. H. (1983), 'Inductive inference: Theory and methods', *ACM Computing Surveys* **15**(3), 237–269.

Baldi, P., Chauvin, Y., Hunkapiller, T. & McClure, M. A. (1993), 'Hidden Markov Models in molecular biology: New algorithms and applications', this volume.

Baum, L. E., Petrie, T., Soules, G. & Weiss, N. (1970), 'A maximization technique occuring in the statistical analysis of probabilistic functions in Markov chains', *The Annals of Mathematical Statistics* **41**(1), 164–171.

Berger, J. O. (1985), *Statistical Decision Theory and Bayesian Analysis*, Springer Verlag, New York.

Buntine, W. (1992), Learning classification trees, *in* D. J. Hand, ed., 'Artificial Intelligence Frontiers in Statistics: AI and Statistics III', Chapman & Hall.

Haussler, D., Krogh, A., Mian, I. S. & Sjölander, K. (1992), Protein modeling using hidden Markov models: Analysis of globins, Technical Report UCSC-CRL-92-23, Computer and Information Sciences, University of California, Santa Cruz, Ca. Revised Sept. 1992.

Hopcroft, J. E. & Ullman, J. D. (1979), *Introduction to Automata Theory, Languages, and Computation*, Addison-Wesley, Reading, Mass.

Horning, J. J. (1969), A study of grammatical inference, Technical Report CS 139, Computer Science Department, Stanford University, Stanford, Ca.

Lari, K. & Young, S. J. (1990), 'The estimation of stochastic context-free grammars using the Inside-Outside algorithm', *Computer Speech and Language* **4**, 35–56.

MacKay, D. J. C. (1992), 'Bayesian interpolation', *Neural Computation* **4**, 415–447.

Omohundro, S. M. (1992), Best-first model merging for dynamic learning and recognition, Technical Report TR-92-004, International Computer Science Institute, Berkeley, Ca.

Porat, S. & Feldman, J. A. (1991), 'Learning automata from ordered examples', *Machine Learning* **7**, 109–138.

Rabiner, L. R. & Juang, B. H. (1986), 'An introduction to Hidden Markov Models', *IEEE ASSP Magazine* **3**(1), 4–16.

Rissanen, J. (1983), 'A universal prior for integers and estimation by minimum description length', *The Annals of Statistics* **11**(2), 416–431.

Stolcke, A. & Omohundro, S. (1993), Best-first model merging for Hidden Markov Model induction, Technical Report TR-93-003, International Computer Science Institute, Berkeley, Ca.

Tomita, M. (1982), Dynamic construction of finite automata from examples using hill-climbing, *in* 'Proceedings of the 4th Annual Conference of the Cognitive Science Society', Ann Arbor, Mich., pp. 105–108.

Wallace, C. S. & Freeman, P. R. (1987), 'Estimation and inference by compact coding', *Journal of the Royal Statistical Society, Series B* **49**(3), 240–265.